# Help or Hinder: Bayesian Models of Social Goal Inference

**Tomer D. Ullman, Chris L. Baker, Owen Macindoe, Owain Evans,**
**Noah D. Goodman and Joshua B. Tenenbaum**
{tomeru, clbaker, owenm, owain, ndg, jbt}@mit.edu
Department of Brain and Cognitive Sciences
Massachusetts Institute of Technology

## Abstract

Everyday social interactions are heavily influenced by our snap judgments about others' goals. Even young infants can infer the goals of intentional agents from observing how they interact with objects and other agents in their environment: e.g., that one agent is 'helping' or 'hindering' another's attempt to get up a hill or open a box. We propose a model for how people can infer these social goals from actions, based on inverse planning in multiagent Markov decision problems (MDPs). The model infers the goal most likely to be driving an agent's behavior by assuming the agent acts approximately rationally given environmental constraints and its model of other agents present. We also present behavioral evidence in support of this model over a simpler, perceptual cue-based alternative.

## 1   Introduction

Humans make rapid, consistent intuitive inferences about the goals of agents from the most impoverished of visual stimuli. On viewing a short video of geometric shapes moving in a 2D world, adults spontaneously attribute to them an array of goals and intentions [7]. Some of these goals are simple, e.g. reaching an object at a particular location. Yet people also attribute complex *social* goals, such as helping, hindering or protecting another agent. Recent studies suggest that infants as young as six months make the same sort of complex social goal attributions on observing simple displays of moving shapes, or (at older ages) in displays of puppets interacting [6].

How do humans make these rapid social goal inferences from such impoverished displays? On one approach, social goals are inferred directly from perceptual cues in a bottom-up fashion. For example, infants in [6] may judge that a triangle pushing a circle up a hill is helping the circle get to the top of the hill simply because the circle is moving the triangle in the direction the triangle was last observed moving on its own. This approach, which has been developed by Blythe et al. [3], seems suited to explain the rapidity of goal attribution, without the need for mediation from higher cognition. On an alternative approach, these inferences come from a more cognitive and top-down system for goal attribution. The inferences are based not just on perceptual evidence, but also on an intuitive theory of mind on which behavior results from rational plans in pursuit of goals. On this approach, the triangle is judged to be helping the circle because in some sense he *knows* what the circle's goal is, *desires* for the circle to achieve the goal, constructs a rational plan of action that he *expects* will increase the probability of the circle realizing the goal. The virtue of this theory-of-mind approach is its generality, accounting for a much wider range of social goal inferences that cannot be reduced to simple perceptual cues. Our question here is whether the rapid goal inferences we make in everyday social situations, and that both infants and adults have been shown to make from simple perceptual displays, require the sophistication of a theory-based approach or can be sufficiently explained in terms of perceptual cues.

This paper develops the theory-based approach to intuitive social goal inference. There are two main challenges for this approach. The first is to formalize social goals (e.g. helping or hindering) and to incorporate this formalization into a general computational framework for goal inference that is based on theory of mind. This framework should enable the inference that agent A is helping or hindering agent B from a joint goal inference based on observing A and B interacting. Inference should be possible even with minimal prior knowledge about the agents and without knowledge of B's goal. The second challenge is to show that this computational model provides a qualitative and quantitative fit to rapid human goal inferences from dynamic visual displays. Can inference based on abstract criteria for goal attribution that draws on unobservable mental states (e.g. beliefs, goals, planning abilities) explain fast human judgments from impoverished and unfamiliar stimuli?

In addressing the challenge of formalization, we present a formal account of social goal attribution based on the abstract criterion of A helping (or hindering) B by acting to maximize (minimize) Bs probability of realizing his goals. On this account, agent A rationally maximizes utility by maximizing (minimizing) the expected utility of B, where this expectation comes from As model of Bs goals and plans of action. We incorporate this formalization of helping and hindering into an existing computational framework for theory-based goal inference, on which goals are inferred from actions by inverting a generative rational planning (MDP) model [1]. The augmented model allows for the inference that A is helping or hindering B from stimuli in which B's goal is not directly observable. We test this Inverse Planning model of social goal attribution on a set of simple 2D displays, comparing its performance to that of an alternative model which makes inferences directly from visual cues, based on previous work such as that of Blythe et al. [3].

## 2 Computational Framework

Our framework assumes that people represent the causal role of agents' goals in terms of an intuitive principle of rationality [4]: the assumption that agents will tend to take efficient actions to achieve their goals, given their beliefs about the world. For agents with simple goals toward objects or states of the world, the principle of rationality can be formalized as probabilistic planning in Markov decision problems (MDPs), and previous work has successfully applied inverse planning in MDPs to explain human inferences about the object-directed goals of maze-world agents [2]. Inferences of simple relational goals between agents (such as chasing and fleeing) from maze-world interactions were considered by Baker, Goodman and Tenenbaum [1], using multiagent MDP-based inverse planning. In this paper, we present a framework for modeling inferences of more complex social goals, such as helping and hindering, where an agent's goals depend on the goals of other agents.

We will define two types of agents: simple agents, which have object-directed goals and do not represent other agents' goals, and complex agents, which have either social or object-directed goals, and represent other agents' goals and reason about their likely behavior. For each type of agent and goal, we describe the multiagent MDPs they define. We then describe joint inferences of object-directed and social goals based on the Bayesian inversion of MDP models of behavior.

### 2.1 Planning in multiagent MDPs

An MDP $\mathcal{M} = (\mathcal{S}, \mathcal{A}, \mathcal{T}, \mathcal{R}, \gamma)$ is a tuple that defines a model of an agent's planning process. $\mathcal{S}$ is an encoding of the world into a finite set of mutually exclusive *states*, which specifies the set of possible configurations of all agents and objects. $\mathcal{A}$ is the set of actions and $\mathcal{T}$ is the transition function, which encodes the physical laws of the world, i.e. $\mathcal{T}(S_{t+1}, S_t, A_t) = P(S_{t+1}|S_t, A_t)$ is the marginal distribution over the next state, given the current state and the agent's action (marginalizing over all other agents' actions). $\mathcal{R} : \mathcal{S} \times \mathcal{A} \to \mathbb{R}$ is the reward function, which provides agents with real-valued rewards for each state-action pair, and $\gamma$ is the discount factor. The following subsections will describe how $\mathcal{R}$ depends on the agent's goal $G$ (object-directed or social), and how $\mathcal{T}$ depends on the agent's type (simple or complex). We then describe how agents plan over multiagent MDPs.

#### 2.1.1 Reward functions

**Object-directed rewards** The reward function induced by an object-directed goal $G$ is straightforward. We assume that $\mathcal{R}$ is an additive function of state rewards and action costs, such that $\mathcal{R}(S, A) = r(S) - c(S, A)$. We consider a two-parameter family of reward functions, parameterized

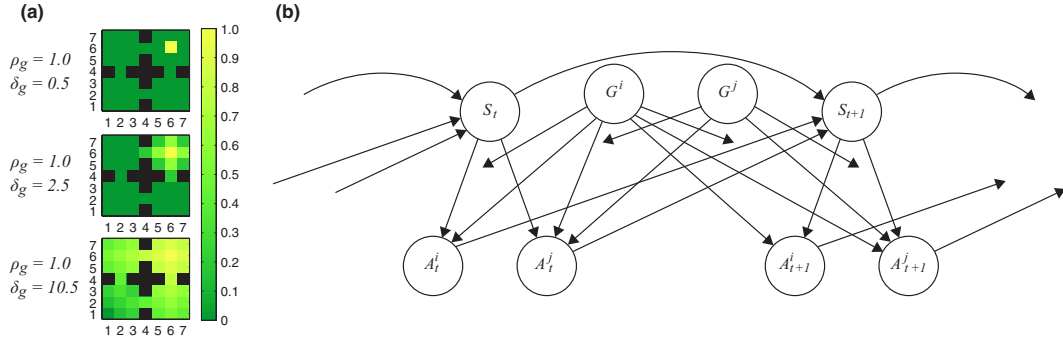

Figure 1: **(a)** Illustration of the state reward functions from the family defined by the parameters $\rho_g$ and $\delta_g$. The agent's goal is at (6,6), where the state reward is equal to $\rho_g$. The state reward functions range from a unit reward in the goal location (row 1) to a field of reward that extends to every location in the grid (row 3). **(b)** Bayes net generated by multiagent planning. In this figure, we assume that there are two agents, $i$ and $j$, with $i$ simple and $j$ complex. The parameters $\{\rho_g^i, \delta_g^i, \rho_o^i, \rho_g^j, \delta_g^j\}$ and $\beta$ are omitted from the graphical model for readability.

by $\rho_g$ and $\delta_g$, which captures the intuition that different kinds of object goals induce different rewards in space. For instance, on a hot summer day in the park, a drinking fountain is only rewarding when one is standing directly next to it. In contrast, a flower's beauty is greatest from up close, but can also be experienced from a range of distances and perspectives. Specifically, $\rho_g$ and $\delta_g$ determine the scale and shape of the state reward function, with $r^i(S) = \max(\rho_g(1 - \text{distance}(S, i, G)/\delta_g), 0)$, where $\text{distance}(S, i, G)$ is the geodesic distance between agent $i$ and the goal. With $\delta_g \leq 1$, the reward function has a unit value of $r(S) = \rho_g$ when the agent and object goal occupy the same location, i.e. when $\text{distance}(S, i, G) = 0$, and $r(S) = 0$ otherwise (see Fig. 1(a), row 1). When $\delta_g > 1$, there is a "field" of positive reward around the goal, with a slope of $-\rho_g/\delta_g$ (see Fig. 1(a), rows 2 and 3). The state reward is maximal at $\text{distance}(S, i, G) = 0$, where $r(S) = \rho_g$, and decreases linearly with the agent's geodesic distance from the goal, reaching a minimum of $r(S) = 0$ when $\text{distance}(S, i, G) \geq \delta_g$.

**Social rewards for helping and hindering**  For complex agent $j$, the state reward function induced by a social goal $G^j$ depends on the cost of $j$'s action $A^j$, as well as the reward function $\mathcal{R}^i$ of the agent that $j$ wants to help or hinder. Specifically, $j$'s reward function is the difference of the expectation of $i$'s reward function and $j$'s action cost function, such that $\mathcal{R}^j(S, A^j) = \rho_o \mathbb{E}_{A^i}[\mathcal{R}^i(S, A^i)] - c(S, A^j)$. $\rho_o$ is the social agent's scaling of the expected reward of state $S$ for agent $i$, which determines how much $j$ "cares" about $i$ relative to its own costs. For helping agents, $\rho_o > 0$, and for hindering agents, $\rho_o < 0$. Computing the expectation $\mathbb{E}_{A^i}[\mathcal{R}^i(S, A^i)]$ relies on the social agent's model of $i$'s planning process, which we will describe below.

### 2.1.2  State-transition functions

In our interactive setting, $\mathcal{T}^i$ depends not just on $i$'s action, but on all other agents' actions as well. Agent $i$ is assumed to compute $\mathcal{T}^i(S_{t+1}, S_t, A_t^i)$ by marginalizing over $A_t^j$ for all $j \neq i$:

$$\mathcal{T}^i(S_{t+1}, S_t, A_t^i) = P(S_{t+1}|S_t, A_t^i) = \sum_{A_t^{j \neq i}} P(S_{t+1}|S_t, A_t^{1:n}) \prod_j P(A_t^j \in A_t^{j \neq i}|S_t, G^{1:n})$$

where $n$ is the number of agents. This computation requires that an agent have a model of all other agents, whether simple or complex.

**Simple agents**  We assume that the simple agents model other agents as randomly selecting actions in proportion to the softmax of their expected cost, i.e. for agent $j$, $P(A^j|S) \propto \exp(\beta \cdot c(S, A^j))$.

**Complex agents**  We assume that the social agent $j$ uses its model of other agents' planning process to compute $P(A^i|S, G^i)$, for $i \neq j$, allowing for accurate prediction of other agents' actions.

We assume agents have access to the true environment dynamics. This is a simplification of a more realistic framework in which agents have only partial or false knowledge about the environment.

### 2.1.3 Multiagent planning

Given the variables of MDP $M$, we can compute the optimal state-action value function $\mathcal{Q}^*$ : $\mathcal{S} \times \mathcal{A} \to \mathbb{R}$, which determines the expected infinite-horizon reward of taking an action in each state. We assume that agents have softmax-optimal policies, such that $P(A|S, G) \propto \exp(\beta \mathcal{Q}^*(S, A))$, allowing occasional deviations from the optimal action depending on the parameter $\beta$, which determines agents' level of determinism (higher $\beta$ implies higher determinism, or less randomness). In a multiagent setting, joint value functions can be optimized recursively, with one agent representing the value function of the other, and the other representing the representation of the first, and so on to an arbitrarily high order [10]. Here, we restrict ourselves to the first level of this reasoning hierarchy. That is, an agent A can at most represent an agent B's reasoning about A's goals and actions, but not a deeper recursion in which B reasons about A reasoning about B.

### 2.2 Inverse planning in multiagent MDPs

Once we have computed $P(A^i|S, G^i)$ for agents 1 through $n$ using multiagent planning, we use Bayesian inverse planning to infer agents' goals, given observations of their behavior. Fig. 1(b) shows the structure of the Bayes net generated by multiagent planning, and over which goal inferences are performed. Let $\theta = \{\rho_g^i, \delta_g^i, \rho_o^i\}^{1:n}$ be a vector of the parameters of the agents' reward functions. We compute the joint posterior marginal of agent $i$'s goal $G^i$ and $\theta$, given the observed state-sequence $S_{1:T}$ and the action-sequences $A_{1:T-1}^{1:n}$ of agents $1{:}n$ using Bayes' rule:

$$P(G^i, \theta | S_{1:T}, A_{1:T-1}^{1:n}, \beta) \propto \sum_{G^{j \neq i}} P(A_{1:T-1}^{1:n} | S_{1:T}, G^{1:n}, \theta, \beta) P(G^{1:n}) P(\theta) \tag{1}$$

To generate goal inferences for our experimental stimuli to compare with people's judgments, we integrate Eq. 1 over a range of $\theta$ values for each stimulus trial:

$$P(G^i | S_{1:T}, A_{1:T-1}^{1:n}, \beta) = \sum_{\theta} P(G^i, \theta | S_{1:T}, A_{1:T-1}^{1:n}, \beta) \tag{2}$$

This allows our models to infer the combination of goals and reward functions that best explains the agents' behavior for each stimulus.

## 3 Experiment

We designed an experiment to test the Inverse Planning model of social goal attributions in a simple 2D maze-world domain, inspired by the stimuli of many previous studies involving children and adults [7, 5, 8, 6, 9, 12]. We created a set of videos which depicted agents interacting in a maze. Each video contained one "simple agent" and one "complex agent", as described in the Computational Framework section. Subjects were asked to attribute goals to the agents after viewing brief snippets of these videos. Many of the snippets showed agent behavior consistent with more than one hypothesis about the agents' goals. Data from subjects was compared to the predictions of the Inverse Planning model and a model based on simple visual cues that we describe in the Modeling subsection below.

### 3.1 Participants

Participants were 20 adults, 8 female and 12 male. Mean age was 31 years.

### 3.2 Stimuli

We constructed 24 scenarios in which two agents moved around a 2D maze (shown in Fig. 2). The maze always contained two potential object goals (a flower and a tree), and on 12 of the 24 scenarios it also contained a movable obstacle (a boulder). The scenarios were designed to satisfy two criteria. First, scenarios were to have agents acting in ways that were consistent with more than one hypothesis concerning their goals, with these ambiguities between goals sometimes being resolved as the scenario developed (see Fig. 2(a)). This criterion was included to test our model's predictions based on ambiguous action sequences. Second, scenarios were to involve a variety of

perceptually distinct plans of action that might be interpreted as issuing from helping or hindering goals. For example, one agent pushing another toward an object goal, removing an obstacle from the other agent's path, and moving aside for the other agent (all of which featured in our scenarios) could all be interpreted as helping. This criterion was included to test our formalization of social goals as based on an abstract relation between reward functions. In our model, social agents act to maximize or minimize the reward of the other agent, and the precise manner in which they do so will vary depending on the structure of the environment and their initial positions.

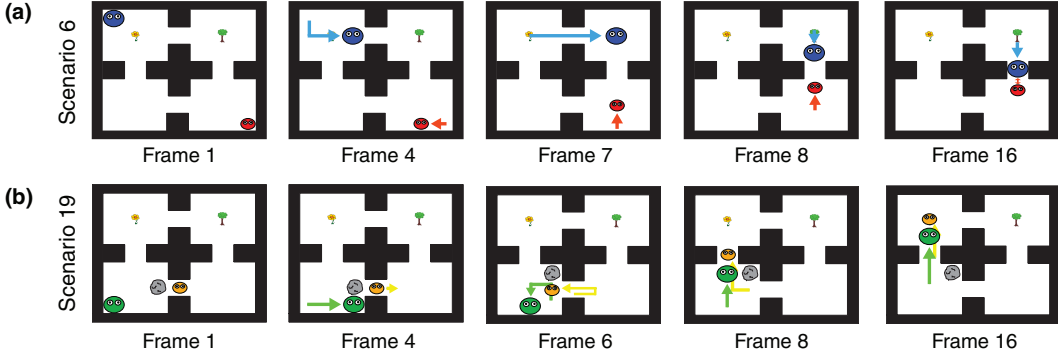

Figure 2: Example interactions between Small and Large agents. Agents start as in Frame 1 and progress through the sequence along the corresponding colored paths. Each frame after Frame 1 corresponds to a *probe point* at which the video was cut off and subjects were asked to judge the agents' goals. **(a)** The Large agent moves over each of the goal objects (Frames 1-7) and so the video is initially ambiguous between his having an object goal and a social goal. Disambiguation occurs from Frame 8, when the Large agent moves down and blocks the Small agent from continuing his path up to the object goal. **(b)** The Large agent moves the boulder, unblocking the Small agent's shortest path to the flower (Frames 1-6). Once the Small agent moves into the same room (6), the Large agent pushes him onto the flower and allows him to rest there (8-16).

Each scenario featured two different agents, which we call "Small" and "Large". Large agents were visually bigger and are able to shift both movable obstacles and Small agents by moving directly into them. Large agents never fail in their actions, e.g. when they try to move left, they indeed move left. Small agents were visually smaller, and could not shift agents or boulders. In our scenarios, the actions of Small agents failed with a probability of about 0.4. Large agents correspond to the "complex agents" introduced in Section 2, in that they could have either object-directed goals or social goals (helping or hindering the Small agent). Small agents correspond to "simple agents" and could have only object goals.

We produced videos of 16 frames in length, displaying each scenario. We showed three snippets from each video, which stopped some number of frames before the end. For example, the three snippets of scenario 6 were cut off at frames 4, 7, and 8 respectively (see Fig. 2(a)). Subjects were asked to make goal attributions at the end of both the snippets and the full 16-frame videos. Asking subjects for goal attributions at multiple points in a sequence allowed us to track the change in their judgments as evidence for particular goals accumulated. These cut-off or *probe* points were selected to try to capture key events in the scenarios and so occurred before and after crucial actions that disambiguated between different goals. Since each scenario was used to create 4 stimuli of varying length, there was a total of 96 stimuli.

### 3.3 Procedure

Subjects were initially shown a set of familiarization videos of agents interacting in the maze, illustrating the structural properties of the maze-world e.g. the actions available to agents and the possibility of moving obstacles) and the differences between Small and Large agents. The experimental stimuli were then presented in four blocks, each containing 24 videos. Scenarios were randomized within blocks across subjects. The left-right orientation of agents and goals was counterbalanced across subjects. Subjects were told that each snippet would contain two new agents (one Small and one Large) and this was highlighted in the stimuli by randomly varying the color of the agents for each snippet. Subjects were told that agents had complete knowledge of the physical structure of the maze, including the position of all goals, agents and obstacles. After each snippet, subjects made

a forced-choice for the goal of each agent. For the Large agent, they could select either of the two social goals and either of the two object goals. For the Small agent, they could choose only from the object goals. Subjects also rated their confidence on a 3-point scale.

## 3.4 Modeling

Model predictions were generated using Eq. 2, assuming uniform priors on goals, and were compared directly to subjects' judgments. In our experiments, the world was given by a 2D maze-world, and the state space included the set of positions that agents and objects can jointly occupy without overlapping. The set of actions included $Up$, $Down$, $Left$, $Right$ and $Stay$ and we assume that $c(S, A \in \{Up, Down, Left, Right\}) = 1$, and $c(S, Stay) = 0.1$ to reflect the greater cost of moving than staying put. We set $\beta$ to 2 and $\gamma$ to 0.99, following [2].

For the other parameters (namely $\rho_g$, $\delta_g$ and $\rho_o$) we integrated over a range of values that provided a good statistical fit to our stimuli. For instance, some stimuli were suggestive of "field" goals rather than point goals, and marginalizing over $\delta_g$ allowed our model to capture this. Values for $\rho_g$ ranged from 0.5 to 2.5, going from a weak to a strong reward. For $\delta_g$ we integrated over three possible values: 0.5, 2.5 and 10.5. These corresponded to "point" object goals (agent receives reward for being on the goal only), "room" object goals (agent receives the most reward for being on the goal and some reward for being in the same room as the goal) and "full space" object goals (agent receives reward at any point in proportion to distance from goal). Values for $\rho_o$ ranged from 1 to 9, from caring weakly about the other agent to caring about it to a high degree.

We compared the Inverse Planning model to a model that made inferences about goals based on simple visual cues, inspired by previous heuristic- or perceptually-based accounts of human action understanding of similar 2D animated displays [3, 11]. Our aim was to test whether accurate goal inferences could be made simply by recognizing perceptual cues that correlate with goals, rather than by inverting a rational model. We constructed our "Cue-based" model by selecting ten visual cues (listed below), including nearly all the applicable cues from the existing cue-based model described in [3], leaving out those that do not apply to our stimuli, such as heading, angle and acceleration. We then formulated an inference model based on these cues by using multinomial logistic regression to subjects' average judgments. The set of cues was as following: (1) the distance moved on the last timestep, (2) the change in movement distance between successive timesteps, (3+4) the geodesic distance to goals 1 and 2, (5+6) the change in distance to goals 1 and 2 (7) the distance to Small, (8) the change in distance to Small, (9+10) the distance of Small to goals 1 and 2.

## 3.5 Results

Because our main interest is in judgments about the social goals of representationally complex agents, we analzyed only subjects' judgments about the Large agents. Each subject judged a total of 96 stimuli, corresponding to 4 time points along each of 24 scenarios. For each of these 96 stimuli, we computed an empirical probability distribution representing how likely a subject was to believe that the Large agent had each of the four goals 'flower', 'tree', 'help', or 'hinder', by averaging judgments for that stimulus across subjects, weighted by subjects' confidence ratings. All analyses then compared these average human judgments to the predictions of the Inverse Planning and Cue-based models.

Across all goal types, the overall linear correlations between human judgments and predictions from the two models appear similar: $r = 0.83$ for the Inverse Planning model, and $r = 0.77$ for the Cue-based model. Fig. 3 shows these correlations broken down by goal type, and reveals significant differences between the models on social versus object goals. The Inverse Planning model correlates well with judgments for all goal types: $r = 0.79, 0.77, 0.86, 0.81$ for flower, tree, helping, and hindering respectively. The Cue-based model correlates well with judgments for object goals ($r = 0.85, 0.90$ for flower, tree) – indeed slightly better the Inverse Planning model – but much less well for social goals ($r = 0.67, 0.66$ for helping, hindering). The most notable differences come on the left-hand sides of the bottom panels in Fig. 3. There are many stimuli for which people are very confident that the Large agent is either helping or hindering, and the Inverse Planning model is similarly confident (bar heights near 1). The Cue-based model, in contrast, is unsure: it assigns roughly equal probabilities of helping or hindering to these cases (bar heights near 0.5). In other words, the Cue-based model is effective at inferring simple object goals of maze-world agents, but

is generally unable to distinguish between the more complex goals of helping and hindering. When constrained to simply differentiating between social and object goals both models succeed equally ($r = 0.84$), where in the Cue-based model this is probably because moving away from the object goals serves as a good cue to separate these categories. However, the Inverse Planning model is more successful in differentiating the right goal within social goals ($r = 0.73$ for the Inverse Planning model vs. $r = 0.44$ for the Cue-based model).

Several other general trends in the results are worth noting. The Inverse Planning model fits very closely with the judgments subjects make after the full 16-frame videos. On 23 of the 24 scenarios, humans and the Inverse Planning model have the highest posterior / rating in the same goal ($r = 0.97$, contrasted with $r = 0.77$ for the Cue-based model). Note that in the one scenario for which humans and the Inverse Planning model disagreed after observing the full sequence, both humans and the model were close to being ambivalent whether the Large agent was hindering or interested in the flower. There is also evidence that the reasonably good overall correlation for the Cue-based model is partially due to overfitting; this should not be surprising given how many free parameters the model has. We divided scenarios into two groups depending on whether a boulder was moved around in the scenario, as movable boulders increase the range of variability in helping and hindering action sequences. When trained on the 'no boulder' cases, the Cue-based model correlates poorly with subjects' average judgments on the 'boulder' cases: $r = 0.42$. The same failure of transfer occurs when the Cue-based model is trained on the 'boulder' cases and tested on the 'no boulder' cases: $r = 0.36$. This is consistent with our general concern that a Cue-based model incorporating many free parameters may do well when tailored to a particular environment, but is not likely to generalize well to new environments. In contrast, the Inverse Planning model captures abstract relations between the agents and their possible goal and so lends itself to a variety of environments.

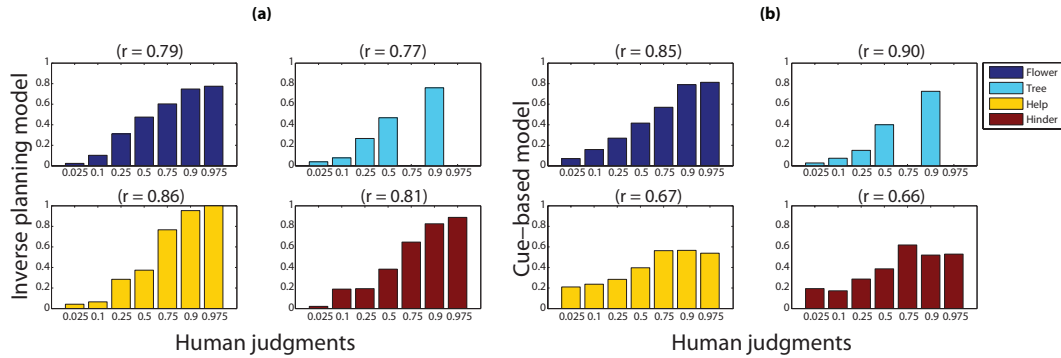

Figure 3: Correlations between human goal judgments and predictions of the Inverse Planning model **(a)** and the Cue-based model **(b)**, broken down by goal type. Bars correspond to bins of stimuli (out of 96 total) on which the average human judgment for the probability of that goal was within a particular range; the midpoint of each bin's range is shown on the x-axis labels. The height of each bar shows the model's average probability judgment for all stimuli in that bin. Linear correlations between the model's goal probabilities and average human judgments for all 96 stimuli are given in the y-axis labels.

The inability of the heuristic model to distinguish between helping and hindering is illustrated by the plots in Fig. 4. In contrast, both the Inverse Planning model and the human subjects are often very confident that an agent is helping and not hindering (or vice versa).

Fig. 4 also illustrates a more general finding, that the Inverse Planning model captures most of the major qualitative shifts (e.g. shifts resulting from disambiguating sequences) in subjects' goal attribution. Figure 4 displays mean human judgments on four scenarios. Probe points (i.e. points within the sequences at which subjects made judgments) are indicated on the plots and human data is compared with predictions from the Inverse Planning model and the Cue-based model.

On scenario 6 (depicted in Fig. 2(a) but with goals switched), both the Inverse Planning model and humans subjects recognize the movement of the Large agent one step off the flower (or the tree in Fig. 2(b)) as strong evidence that Large has a hindering goal. The Cue-based model responds in the same way but with much less confidence in hindering. Even after 8 subsequent frames of action it is unable to decide in favor of hindering over helping.

While the Inverse Planning model and subjects almost always agree by the end of a sequence, they sometimes disagree at early probe points. In scenario 5, both agents start off in the bottom-left room, but with the Small agent right at the entrance to the top-left room. As the Small agent tries to move towards the flower (the top-left goal), the Large agent moves up from below and pushes Small one step towards the flower before moving off to the right to the tree. People interpret the Large agent's action as strong evidence for helping, in contrast with the Inverse Planning model. For the model, because Small is so close to his goal, Large could just as well stay put and save his own action costs. Therefore his movement upwards is not evidence of helping.

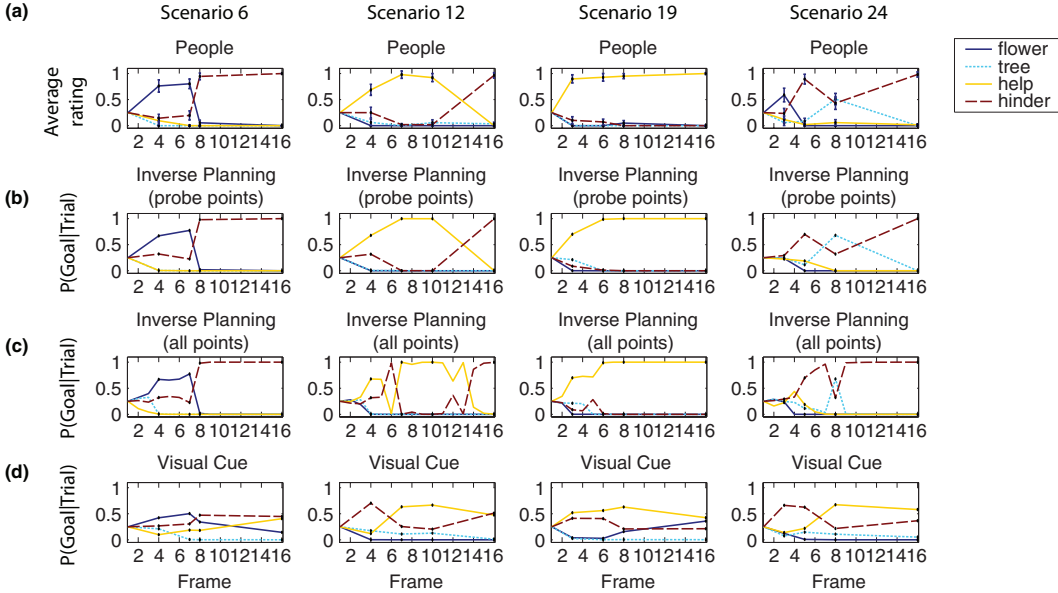

Figure 4: Example data and model predictions. Probe points are marked as black circles. **(a)** Average subject ratings with standard error bars. **(b)** Predictions of Inverse Planning model interpolated from cut points. **(c)** Predictions of Inverse Planning model for all points in the sequence. **(d)** Predictions of Cue-based model.

## 4   Conclusion

Our goal in this paper was to address two challenges. The first was to provide a formalization of social goal attribution incorporated into a general theory-based model for goal attribution. This model had to enable the inference that A is helping or hindering B from interactions between A and B but without prior knowledge of either agent's goal, and to account for the range of behaviors that humans judge as evidence of helping or hindering. The second challenge was for the model to perform well on a demanding inference task in which social goals must be inferred from very few observations without directly observable evidence of agents' goals.

The experimental results presented here go some way to meeting these challenges. The Inverse Planning model classified a diverse range of agent interactions as helping or hindering in line with human judgments. This model also distinguished itself against a model based solely on simple perceptual cues. It produced a closer fit to humans for both social and nonsocial goal attributions, and was far superior to the visual cue model in discriminating between helping and hindering.

These results suggest various lines of further research. One task is to augment this formal model of helping and hindering to capture more of the complexity behind human judgments. On the Inverse Planning model, A will act to advance B's progress only if there is some chance of B actually receiving a nontrivial amount of reward in a future state. However, people often help others towards a goal even if they think it very unlikely that the goal will be achieved. This aspect of helping could be explored by supposing that the utility of a helping agent depends not just on another agent's reward function but also his value function.

**Acknowledgments:** This work was supported by the James S. McDonnell Foundation Causal Learning Collaborative Initiative, ARO MURI grant W911NF-08-1-0242, AFOSR MURI grant FA9550-07-1-0075 and the NSF Graduate Fellowship (CLB).

# References

[1] C. L. Baker, N. D. Goodman, and J. B. Tenenbaum. Theory-based social goal inference. In *Proceedings of the Thirtieth Annual Conference of the Cognitive Science Society*, 2008.

[2] C. L. Baker, J. B. Tenenbaum, and R. R. Saxe. Bayesian models of human action understanding. In *Advances in Neural Information Processing Systems*, volume 18, pages 99–106, 2006.

[3] P. W. Blythe, P. M. Todd, and G. F. Miller. How motion reveals intention: categorizing social interactions. In G. Gigerenzer, P. M. Todd, and the ABC Research Group, editors, *Simple heuristics that make us smart*, pages 257–286. Oxford University Press, New York, 1999.

[4] D. C. Dennett. *The Intentional Stance*. MIT Press, Cambridge, MA, 1987.

[5] G. Gergely, Z. Nádasdy, G. Csibra, and S. Biró. Taking the intentional stance at 12 months of age. *Cognition*, 56:165–193, 1995.

[6] J. K. Hamlin, Karen Wynn, and Paul Bloom. Social evaluation by preverbal infants. *Nature*, 450:557–560, 2007.

[7] F. Heider and M. A. Simmel. An experimental study of apparent behavior. *American Journal of Psychology*, 57:243–249, 1944.

[8] V. Kuhlmeier, Karen Wynn, and Paul Bloom. Attribution of dispositional states by 12-month-olds. *Psychological Science*, 14(5):402–408, 2003.

[9] J. Schultz, K. Friston, D. M. Wolpert, and C. D. Frith. Activation in posterior superior temporal sulcus parallels parameter inducing the percept of animacy. *Neuron*, 45:625–635, 2005.

[10] Wako Yoshida, Ray J. Dolan, and Karl J. Friston. Game theory of mind. *PLoS Computational Biology*, 4(12):1–14, 2008.

[11] Jeffrey M. Zacks. Using movement and intentions to understand simple events. *Cognitive Science*, 28:979–1008, 2004.

[12] P. D. Tremoulet and J. Feldman The influence of spatial context and the role of intentionality in the interpretation of animacy from motion. *Perception and Psychophysics*, 29:943–951, 2006.

